# Smoothing Regularizers for Projective Basis Function Networks

**John E. Moody** and **Thorsteinn S. Rögnvaldsson** *
Department of Computer Science, Oregon Graduate Institute
PO Box 91000, Portland, OR 97291
moody@cse.ogi.edu    denni@cca.hh.se

## Abstract

Smoothing regularizers for radial basis functions have been studied extensively, but no general smoothing regularizers for *projective basis functions* (PBFs), such as the widely-used sigmoidal PBFs, have heretofore been proposed. We derive new classes of algebraically-simple $m^{th}$-order smoothing regularizers for networks of the form $f(W, x) = \sum_{j=1}^{N} u_j g \left[ x^T v_j + v_{j0} \right] + u_0$, with general projective basis functions $g[\cdot]$. These regularizers are:

$$R_G(W, m) = \sum_{j=1}^{N} u_j^2 \|v_j\|^{2m-1} \quad \text{Global Form}$$

$$R_L(W, m) = \sum_{j=1}^{N} u_j^2 \|v_j\|^{2m} \quad \text{Local Form}$$

These regularizers bound the corresponding $m^{th}$-order smoothing integral

$$S(W, m) = \int d^D x \, \Omega(x) \left\| \frac{\partial^m f(W, x)}{\partial x^m} \right\|^2 \quad ,$$

where $W$ denotes all the network weights $\{u_j, u_0, v_j, v_0\}$, and $\Omega(x)$ is a weighting function on the $D$-dimensional input space. The global and local cases are distinguished by different choices of $\Omega(x)$.

The simple algebraic forms $R(W, m)$ enable the direct enforcement of smoothness without the need for costly Monte-Carlo integrations of $S(W, m)$. The new regularizers are shown to yield better generalization errors than weight decay when the implicit assumptions in the latter are wrong. Unlike weight decay, the new regularizers distinguish between the roles of the input and output weights and capture the interactions between them.

# 1   Introduction: What are the right biases?

Regularization is a technique for reducing prediction risk by balancing model bias and model variance. A regularizer $R(W)$ imposes prior constraints on the network parameters $W$. Using squared error as the most common example, the objective functional that is minimized during training is

$$E = \frac{1}{2M} \sum_{i=1}^{M} [y^{(i)} - f(W, x^{(i)})]^2 + \lambda R(W) \; , \tag{1}$$

where $y^{(i)}$ are target values corresponding to the inputs $x^{(i)}$, $M$ is the number of training patterns, and the regularization parameter $\lambda$ controls the importance of the prior constraints relative to the fit to the data. Several approaches can be applied to estimate $\lambda$ (e.g. Eubank (1988) or Wahba (1990)).

Regularization reduces model variance at the cost of some model bias. An important question arises: What are the right biases? (Geman, Bienenstock & Doursat 1992). A good choice of $R(W)$ will result in lower expected prediction error than will a poor choice.

Weight decay is often used effectively, but it is an *ad hoc* technique that controls weight values without regard to the function $f(\cdot)$. It is thus not necessarily optimal and not appropriate for arbitrary function parameterizations. It will give very different results, depending upon whether a function is parameterized, for example, as $f(w, x)$ or as $f(w^{-1}, x)$.

Since many real world problems are intrinsically smooth, we propose that in many cases, an appropriate bias to impose is to favor solutions with low $m^{th}$-order curvature. Direct penalization of curvature is a parametrization-independent approach. The desired regularizer is the standard $D$ dimensional curvature functional of order $m$:

$$S(W, m) = \int d^D x \, \Omega(x) \left\| \frac{\partial^m f(W, x)}{\partial x^m} \right\|^2 \; . \tag{2}$$

Here $\| \; \|$ denotes the ordinary euclidean tensor norm and $\partial^m / \partial x^m$ denotes the $m^{th}$ order differential operator. The weighting function $\Omega(x)$ ensures that the integral converges and determines the region over which we require the function to be smooth. $\Omega(x)$ is not required to be equal to the input density $p(x)$, and will most often be different.

The use of smoothing functionals like (2) has been extensively studied for smoothing splines (Eubank 1988, Hastie & Tibshirani 1990, Wahba 1990) and for radial basis function (RBF) networks (Powell 1987, Poggio & Girosi 1990, Girosi, Jones & Poggio 1995). However, no general class of smoothing regularizers that directly enforce smoothness $S(W, m)$ for *projective basis functions* (PBFs), such as the widely used sigmoidal PBFs, has been previously proposed.

Since explicit enforcement of smoothness using (2) requires costly, impractical Monte-Carlo integrations,[1] we derive algebraically-simple regularizers $R(W, m)$ that tightly bound $S(W, m)$.

# 2   Derivation of Simple Regularizers from Smoothing Functionals

We consider single hidden layer networks with $D$ input variables, $N_h$ nonlinear hidden units, and $N_o$ linear output units. For clarity, we set $N_o = 1$, and drop the subscript on $N_h$

(the derivation is trivially extended to the case $N_o > 1$). Thus, our network function is

$$f(x) = \sum_{j=1}^{N} u_j g[\theta_j, x] + u_0 \tag{3}$$

where $g[\cdot]$ are the nonlinear transfer functions of the internal hidden units, $x \in R^D$ is the input vector[2] , $\theta_j$ are the parameters associated with internal unit $j$, and $W$ denotes all parameters in the network.

For regularizers $R(W)$, we will derive *strict upper bounds* for $S(W, m)$. We desire the regularizers to be as general as possible so that they can easily be applied to different network models. Without making any assumptions about $\Omega(x)$ or $g(\cdot)$, we have the upper bound

$$S(W, m) \leq N \sum_{j=1}^{N} u_j^2 \int d^D x \Omega(x) \left\| \frac{\partial^m g[\theta_j, x]}{\partial x^m} \right\|^2, \tag{4}$$

which follows from the inequality $\left( \sum_{i=1}^{N} a_i \right)^2 \leq N \sum_{i=1}^{N} a_i^2$. We consider two possible options for the weighting function $\Omega(x)$. One is to require *global* smoothness, in which case $\Omega(x)$ is a very wide function that covers all relevant parts of the input space (e.g. a very wide gaussian distribution or a constant distribution). The other option is to require *local* smoothness, in which case $\Omega(x)$ approaches zero outside small regions around some reference points (e.g. the training data).

### 2.1 Projective Basis Representations

Projective basis functions (PBFs) are of the form $g[\theta_j, x] = g\left[ x^T v_j + v_{j0} \right]$, where $\theta_j = \{v_j, v_{j0}\}$, $v_j = (v_{j1}, v_{j2}, \ldots, v_{jD})$ is the vector of weights connecting hidden unit $j$ to the inputs, and $v_{j0}$ is the bias, offset, or threshold. For PBFs, expression (4) simplifies to

$$S(W, m) \leq N \sum_{j=1}^{N} u_j^2 \|v_j\|^{2m} I_j(W, m), \tag{5}$$

with

$$I_j(W, m) \equiv \int d^D x \Omega(x) \left( \frac{d^m g[z_j(x)]}{dz_j^m} \right)^2 \tag{6}$$

where $z_j(x) \equiv x^T v_j + v_{j0}$.

Although the most commonly used $g[\cdot]$'s are sigmoids, our analysis applies to many other forms, for example flexible fourier units, polynomials, and rational functions.[3] The classes of PBF transfer functions $g[\cdot]$ that are applicable (as determined by $\Omega(x)$) are those for which the integral (8) is finite and well-defined.

### 2.2 Global weighting

For the global case, we select a gaussian form for the weighting function

$$\Omega_G(x) = (\sqrt{2\pi}\sigma)^{-D} \exp\left[ \frac{-\|x\|^2}{2\sigma^2} \right] \tag{7}$$

and require $\sigma$ to be large. Integrating out all dimensions, except the one associated with the projection vector $v_j$, we are left with

$$I_j(W, m) = \frac{1}{\sigma\sqrt{2\pi}} \int_{-\infty}^{\infty} dx e^{-x^2/2\sigma^2} \left( \frac{d^m g[z_j(x)]}{dz_j^m} \right)^2 . \tag{8}$$

If $(d^m g[z]/dz^m)^2$ is integrable and approaches zero outside a region that is small compared to $\sigma$, we can bound (8) by setting the exponential equal to unity. This implies

$$I_j(W, m) \leq \frac{I(m)}{\|v_j\|} \quad \text{with} \quad I(m) \equiv \frac{1}{\sigma\sqrt{2\pi}} \int_{-\infty}^{\infty} dz \left( \frac{d^m g[z]}{dz^m} \right)^2 . \tag{9}$$

The bound of equation (5) then becomes

$$S(W, m) \leq NI(m) \sum_{j=1}^{N} u_j^2 \|v_j\|^{2m-1} = NI(m)R_G(W, m) , \tag{10}$$

where the subscript $G$ stands for *global*. Since $\lambda$ absorbs all constant multiplicative factors, we need only weigh $R_G(W, m)$ into the training objective function.

### 2.2.1   Local weighting

For the local case, we consider weighting functions of the general form

$$\Omega_L(x) = \frac{1}{M} \sum_{i=1}^{M} \Omega(x^{(i)}, \sigma) \tag{11}$$

where $x^{(i)}$ are a set of points, and $\Omega(x^{(i)}, \sigma)$ is a function that decays rapidly for large $\|x - x^{(i)}\|$. We require that $\lim_{\sigma\to 0} \Omega(x^{(i)}, \sigma) = \delta(x - x^{(i)})$. Thus, when the $x^{(i)}$ are the training data points, the limiting distribution of (11) is the *empirical distribution*.

In the limit $\sigma \to 0$, equation (5) becomes

$$S(W, m) \leq \frac{N}{M} \sum_{j=1}^{N} u_j^2 \|v_j\|^{2m} \sum_{i=1}^{M} \left( \frac{d^m g[z_j(x^{(i)})]}{dz_j^m} \right)^2 . \tag{12}$$

For the empirical distribution, we could compute the expression within parenthesis in (12) for each input pattern $x^{(i)}$ during training and use it as our regularization cost. This is done by Bishop (1993) for the special case $m = 2$. However, this requires explicit design for each transfer function and becomes increasingly complicated as we go to higher $m$. To construct a simpler and more general form, we instead assume that the $m^{th}$ derivative of $g[\cdot]$ is bounded from above by $C_L(m) \equiv \max_z \left( \frac{d^m g[z]}{dz^m} \right)^2$ .

This gives the bound

$$S(W, m) \leq NC_L(m) \sum_{j=1}^{N} u_j^2 \|v_j\|^{2m} = NC_L(m)R_L(W, m) \tag{13}$$

for the maximum local curvature of the function (the subscript $L$ denotes *local* limit).

## 3 Empirical Example

We have done extensive simulation studies that demonstrate the efficacy of our new regularizers for PBF networks on a variety of problems. An account is given in Moody & Rögnvaldsson (1996). Here, we demonstrate the value of using smoothing regularizers on a simple problem which illustrates a key difference between smoothing and quadratic weight decay, the two dimensional bilinear function

$$t(x_1, x_2) = x_1 x_2. \tag{14}$$

This example was used by Friedman & Stuetzle (1981) to demonstrate projection pursuit regression. It is the simplest function with interactions between input variables.

We fit this function with one hidden layer networks using the $m = \{1, 2, 3\}$ smoothing regularizers, comparing the results with using weight decay. In a large set of experiments, we find that both the global and local smoothing regularizers with $m = 2$ and $m = 3$ outperform weight decay. An example is shown in figure 1. The local $m = 1$ case performs poorly, which is unsurprising, given that the target function is quadratic. Weight decay performs poorly because it lacks any form of interaction between the input layer and output layer weights $v_j$ and $u_j$.

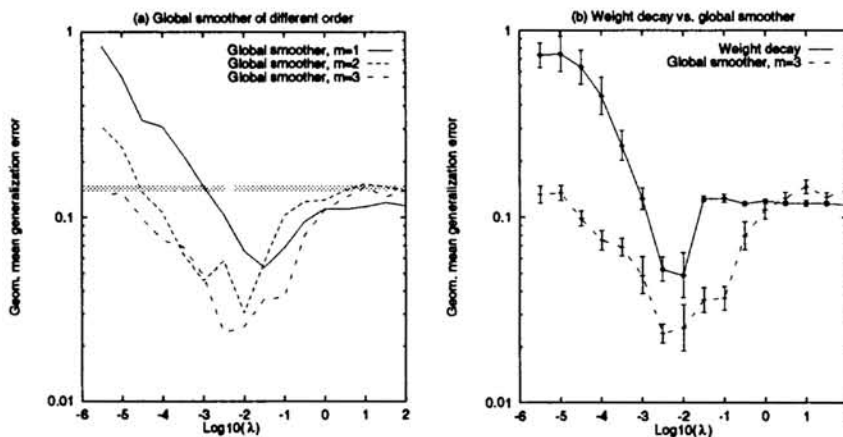

Figure 1: (a) Generalization errors on the $x_1 x_2$ problem, with 40 training data points and a signal-to-noise ratio of 2/3, for different values of the regularization parameter and different orders of the smoothing regularizer. For each value of $\lambda$, 10 networks with 8 hidden units have been trained and averaged (geometric average). The shaded area shows the 95% confidence bands for the average performance of a linear model on the same problem. (b) Similar plot for the $m = 3$ smoother compared to the standard weight decay method. Error bars mark the estimated standard deviation of the mean generalization error of the 10 networks. The $m = 3$ regularizer performs significantly better than weight decay.

## 4 Quality of the Regularizers: Approximations *vs* Bounds

Equations (10) and (13) are strict upper bounds to the smoothness functional $S(W, m)$, eq. (2), in the global and local limits, $\sigma \to \infty$ and $\sigma \to 0$. However, if the bounds are not sufficiently tight, then penalizing $R(W, m)$ may not have the effect of penalizing $S(W, m)$[4].

The bound (4) is tighter the more uncorrelated the $m^{th}$ derivatives of the internal unit activities are. If they are uncorrelated, then the bounds of equations (10) and (13) can be replaced by the approximations:

$$S_G(W,m) \approx I_G(m)R_G(W,m) \tag{15}$$
$$S_L(W,m) \approx C_L(m)R_L(W,m) , \tag{16}$$

using $\left(\sum_{i=1}^{N} a_i\right)^2 \approx \sum_{i=1}^{N} a_i^2$. The right hand sides differ from those in equations (10) and (13) only by a factor of $N$, so these approximations are actually proportional to the bounds.

For our regularizers, the constant factor $N$ doesn't matter, since it can be absorbed into the regularization parameter $\lambda$ (along with the values of the factors $I_G(m)$ or $C_L(m)$). In practical terms, there is no difference between using the upper bounds (10) and (13) or the uncorrelated approximations (15) and (16). Our empirical results (see figure 2) indicate that an approximate linear relationship holds between $S(W,m)$ and $R(W,m)$ for both the global and the local cases. This suggests that the uncorrelated hidden unit assumption yields a good approximation. This approximation also improves with the dimensionality of the input space. Extensive results and discussion are presented in (Moody & Rögnvaldsson 1996).

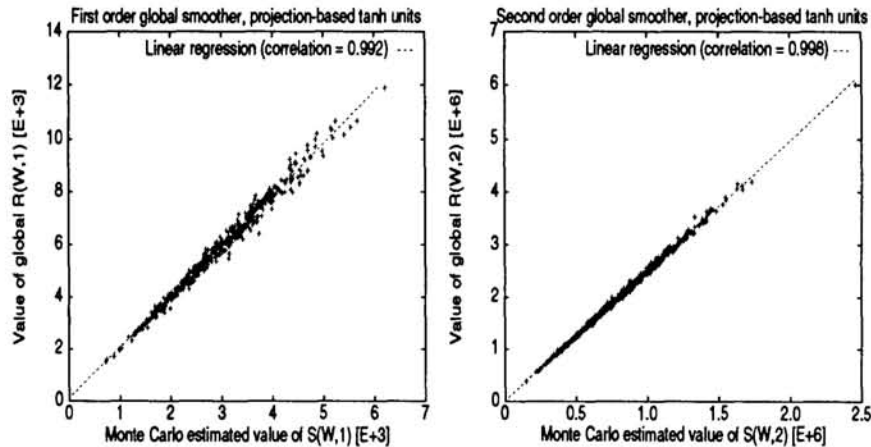

Figure 2: Linear correlation between $S(W,m)$ and the global $R_G(W,m)$ for neural networks with 10 input units, 10 internal $\tanh[\cdot]$ PBF units, and one linear output. The values of $S(W,m)$ are computed through Monte Carlo integration. The left graph shows $m=1$ and the right graph shows $m=2$. Results are similar for the local form $R_L(W,m)$.

## 5  Summary

Our regularizers $R(W,m)$ are the first general class of $m^{th}$–order smoothing regularizers to be proposed for projective basis function (PBF) networks. They apply to large classes of transfer functions $g[\cdot]$, including sigmoids. They differ fundamentally from quadratic weight decay in that they distinguish the roles of the input and output weights and capture the interactions between them.

Our approach is quite different from that developed for smoothing splines and smoothing radial basis functions (RBFs), since we derive smoothing regularizers for given classes of units $g[\theta, x]$, rather than derive the forms of the units $g[\cdot]$ by requiring them to be Greens functions of the smoothing operator $S(\cdot)$. Our approach thus has the advantage that it can be applied to the types of networks most often used in practice, namely PBFs.

In Moody & Rögnvaldsson (1996), we present further analysis and simulation results for PBFs. We have also extended our work to RBFs (Moody & Rögnvaldsson 1997).

## Acknowledgements

Both authors thank Steve Rehfuss and Lizhong Wu for stimulating input. John Moody thanks Volker Tresp for a provocative discussion at a 1991 Neural Networks Workshop sponsored by the Deutsche Informatik Akademie. We gratefully acknowledge support for this work from ARPA and ONR (grant N00014-92-J-4062), NSF (grant CDA-9503968), the Swedish Institute, and the Swedish Research Council for Engineering Sciences (contract TFR-282-95-847).

## Footnotes

*Address as of September 1, 1996: Centre for Computer Architecture, University of Halmstad, P.O.Box 823, S-301 18 Halmstad, Sweden

[1]Note that (2) is not just one integral, but actually $\mathcal{O}(D^m)$ integrals, since the norm of the operator $\partial^m / \partial x^m$ has $\mathcal{O}(D^m)$ terms. This is extremely expensive to compute for large $D$ or large $m$.

[2]Throughout, we use small letter boldface to denote vector quantities.

[3]See for example Moody & Yarvin (1992).

[4]For the proposed regularizers $R(W, m)$ to be effective in penalizing $S(W, m)$, we need only have an approximate monotonic relationship between them.

## References

Bishop, C. (1993), 'Curvature-driven smoothing: A learning algorithm for feedforward networks', *IEEE Trans. Neural Networks* **4**, 882–884.

Eubank, R. L. (1988), *Spline Smoothing and Nonparametric Regression*, Marcel Dekker, Inc.

Friedman, J. H. & Stuetzle, W. (1981), 'Projection pursuit regression', *J. Amer. Stat. Assoc.* **76**(376), 817–823.

Geman, S., Bienenstock, E. & Doursat, R. (1992), 'Neural networks and the bias/variance dilemma', *Neural Computation* **4**(1), 1–58.

Girosi, F., Jones, M. & Poggio, T. (1995), 'Regularization theory and neural network architectures', *Neural Computation* **7**, 219–269.

Hastie, T. J. & Tibshirani, R. J. (1990), *Generalized Additive Models*, Vol. 43 of *Monographs on Statistics and Applied Probability*, Chapman and Hall.

Moody, J. E. & Yarvin, N. (1992), Networks with learned unit response functions, *in* J. E. Moody, S. J. Hanson & R. P. Lippmann, eds, 'Advances in Neural Information Processing Systems 4', Morgan Kaufmann Publishers, San Mateo, CA, pp. 1048–55.

Moody, J. & Rögnvaldsson, T. (1996), Smoothing regularizers for projective basis function networks, Submitted to *Neural Computation*.

Moody, J. & Rögnvaldsson, T. (1997), Smoothing regularizers for radial basis function networks, Manuscript in preparation.

Poggio, T. & Girosi, F. (1990), 'Networks for approximation and learning', *IEEE Proceedings* **78**(9).

Powell, M. (1987), Radial basis functions for multivariable interpolation: a review., *in* J. Mason & M. Cox, eds, 'Algorithms for Approximation', Clarendon Press, Oxford.

Wahba, G. (1990), *Spline models for observational data*, CBMS-NSF Regional Conference Series in Applied Mathematics.